# Reconstructing MEG Sources
# with Unknown Correlations

**Maneesh Sahani**
W. M. Keck Foundation Center
for Integrative Neuroscience,
UC, San Francisco, CA 94143-0732
maneesh@phy.ucsf.edu

**Srikantan S. Nagarajan**
Biomagnetic Imaging Laboratory,
Department of Radiology,
UC, San Francisco, CA 94143-0628
sri@radiology.ucsf.edu

## Abstract

Existing source location and recovery algorithms used in magnetoen-cephalographic imaging generally assume that the source activity at different brain locations is independent or that the correlation structure is known. However, electrophysiological recordings of local field potentials show strong correlations in aggregate activity over significant distances. Indeed, it seems very likely that stimulus-evoked activity would follow strongly correlated time-courses in different brain areas. Here, we present, and validate through simulations, a new approach to source reconstruction in which the correlation between sources is modelled and estimated explicitly by variational Bayesian methods, facilitating accurate recovery of source locations and the time-courses of their activation.

## 1 Introduction

The brain's neuronal activity generates weak magnetic fields (10 fT – 1 pT). Magne-toencephalography (MEG) is a non-invasive technique for detecting and characterising these magnetic fields. MEG sensors use super-conducting quantum interference devices (SQUIDs) to measure the changes in the brain's magnetic field on a millisecond time-scale. When combined with electromagnetic source localisation, magnetic source imaging (MSI) becomes a functional brain imaging method that allows us to characterise macroscopic dynamic neural information processing.

In the past decade, the development of MSI source reconstruction algorithms has progressed significantly [1]. Currently, there are two general approaches to estimating MEG sources: parametric methods and imaging methods [2]. With parametric methods, a few current dipoles of unknown location and moment are assumed to represent the sources of activity in the brain. In this case, solving the inverse problem requires a non-linear optimisation to estimate the position and magnitude of an unknown number of dipoles. With imaging methods, a grid of voxels is used to represent the entire brain volume. The inverse problem is then to recover whole brain activation images, represented by the time-dependent moment and magnitude of an elementary dipole source located at each voxel. This formulation leads to a linear forward model. However, the ill-posed nature of the problem leads to non-unique solutions which must be distinguished by prior information, usually in the form of assumptions regarding the correlation between the sources.

In this paper, we formulate a general spatiotemporal imaging model for MEG data. Our formulation makes no assumptions about the correlation of the sources; instead, we estimate the extent of the correlation by an evidence optimisation procedure within a variational Bayesian framework [3].

## 1.1 MEG imaging

Many standard MEG devices measure the radial gradient of the magnetic field at a number, $d_b$, of sensor locations (typically arranged on a segment of a sphere). Measurements made at a single time can be formed into a $d_b$-dimensional vector **b**; an experiment yields a series of $N$ such samples, giving a $d_b \times N$ data matrix $B$.

This measured field-gradient is affected by a number of different processes. The component we seek to isolate is stimulus- or event-related, and is presumably contributed to by significant activity at a relatively small number of locations in the brain. This **signal** is corrupted by thermal noise at the sensors, and by widespread spontaneous, unrelated brain activity. For our purposes, these are both sources of noise, whose distributions are approximately normal [2] (in the case of the unrelated brain activity, the normality results from the fact that any one sensor sees the sum of effects from a large number of locations). The covariance matrix of this noise, $\Psi$, can be measured approximately by accumulating sensor readings in a quiescent state; simulations suggest that the techniques presented here are reasonably tolerant to mis-estimation of the noise level. Measurements are also affected by other forms of interference associated with experimental electronics or bio-magnetic activity external to the brain. We will not here treat such interference explicitly, instead assuming that major sources have been removed by preprocessing the measured data, *e.g.*, by using blind source separation methods [4].

To represent the significant brain sources, we divide the volume of the brain (or a subsection of that volume that contains the sources) into a number of voxels and then calculate the **lead-field matrix** $L$ that linearly relates the strength of a current dipole in each orientation at each voxel, to the sensor measurements. For simplicity, we assume a spherical volume conductor model, which permits analytical calculation of $L$ independent of the tissue conductivity [2], and which is reasonably accurate for most brain regions [1]. (Non-uniform volume conduction properties of the brain and surrounding tissues can be explicitly accounted for by elaborating the lead-field matrix calculation, but they do not otherwise affect the analysis presented below.) In the simple model, only the two tangential components of the current dipole which fall orthogonal to the radial direction contribute to **b**, and so the source vector **s** has a dimension $d_s$ which is twice the number of voxels $d_v$. The source matrix $S$ associated with the $N$ field measurements has dimensions $d_s \times N$. Thus the probabilistic forward model for MEG measurements is given by

$$\mathbf{b} \sim \mathcal{N}\left(L\mathbf{s}, \Psi\right) \tag{1}$$

Without considerable prior knowledge of the pattern of brain activation, the number of possible degrees of freedom in the source vector, $d_s$, will be far greater than the number of measurements, $d_b$; and so there is no unique maximum-likelihood estimate of **s**. Instead, attempts at source recovery depend, either implicitly or explicitly, on the application of prior knowledge about the source distribution. Most existing methods constrain the source locations and/or activities in various ways: based on anatomical or fMRI data; by maximum entropy, minimum L1 norm, weighted-minimum L2 norm or maximum smoothness priors; or to achieve optimal resolution [1]. Most of these constraints can be formulated as priors for maximum *a posteriori* estimation of the sources (although the original statements do not always make such priors explicit). In addition, some studies have also included temporal constraints on sources such as smoothness or phase-locking between sources [5].

Consider, for example, linear estimates of **s** given by $\hat{\mathbf{s}} = F'\mathbf{b}$. The optimal estimate (in a

least-squares sense) is given by the Wiener filter:

$$F = \langle \mathbf{bb}' \rangle^{-1} \langle \mathbf{bs}' \rangle = \langle \mathbf{bb}' \rangle^{-1} \langle (L\mathbf{s} + \mathbf{n})\mathbf{s}' \rangle = \langle \mathbf{bb}' \rangle^{-1} L \langle \mathbf{ss}' \rangle, \qquad (2)$$

(where $\mathbf{n} \sim \mathcal{N}(0, \Psi)$ is a noise vector uncorrelated with $\mathbf{s}$) and therefore requires knowledge of the source correlation matrix $\langle \mathbf{ss}' \rangle$.

One approach to source reconstruction, the **minimum-variance adaptive beamformer** (or "beamformer" for short), can be viewed as an approximation to the Wiener filter in which the correlation matrix of sensor measurements $\langle \mathbf{bb}' \rangle$ is estimated by the observed correlation $BB'/N$, and the sources at each location are taken to be uncorrelated [6]. If the orientation of each source dipole is known or estimated independently (so that $\mathbf{s}$ contains only one magnitude at each location), then the source correlation matrix $\langle \mathbf{ss}' \rangle$ reduces to a diagonal matrix of gain factors. For the beamformer, these factors are chosen to give a unit "loop gain" for each source i.e. such that $\text{diag}\,[F'L] = \mathbf{1}$. It can be shown that the beamformer only yields accurate results when the number of active sources is few [7]. Thus, this approach makes two assumptions about the sources: an explicit one of decorrelation and an implicit one of sparse activation. Other techniques tend to make similar assumptions. A related algorithm using Multiple Signal Classification (MUSIC) also assumes sparsity and linear independence in the time-series of the sources [1]. Minimum-norm methods can also be viewed as making specific assumptions about the source correlation matrix [8].

In sharp contrast to the assumed independence or known correlation of brain activity in these algorithms, electrophysiological studies have shown pronounced and variable correlations in local potentials measured in different (sometimes widely separated) regions of the brain, and indeed, have argued that these correlations reflect relevant aspects of brain processing [9, 10]. This simple observation has profound consequences for most current MEG imaging algorithms. Not only are they unable to access this source of temporal information about brain function (despite the temporal fidelity of the technique in other respects), but they may also provide inaccurate source localisations or reconstructions by dint of their incorrect assumptions regarding source correlation.

In this paper, we present a novel approach to source reconstruction. Our technique shares with many of the methods described above the assumption of sparsity in source activation. However, it dispenses entirely with assumption of source independence. Instead, we estimate the source correlation matrix from the data by hyperparameter optimisation.

## 2   Model

To parameterise the source correlation matrix in a manner tractable for learning, we assume that the source activities $\mathbf{s}$ are formed by a linear combination, with weight matrix $W$, of $d_z$ independent unit-variance normal **pre-sources z**,

$$\mathbf{s} = W\mathbf{z}; \qquad \mathbf{z} \sim \mathcal{N}(0, I), \qquad (3)$$

so that learning the correlation matrix $\langle \mathbf{ss}' \rangle = WW'$ becomes equivalent to estimation of the weights $W$.[1] The sources are not really expected to have the Gaussian amplitude distribution that this construction implies. Instead, the assumption forms a convenient fiction, making it easy to estimate the source correlation matrix. We show in simulations below that estimation in this framework can indeed yield accurate estimates of the correlation matrix even for non-normally distributed source activity. Once the correlation matrix has been established, estimation using the Wiener filter of (2) provides the best *linear* estimate of source activity (and would be the exact maximum *a posteriori* estimate if the sources really were normally distributed).

The model of (3) parameterises the source correlation in a general way, subject to a maximum rank of $d_z$. This rank constraint does not by itself favour sparsity in the source distribution, and could easily be chosen to be equal to $d_s$. Instead, the sparsity emerges from a hyperparameter optimisation similar to the automatic relevance determination (ARD) of Mackay and Neal [11] (see also [12, 13]). Equation (3) defined a prior on **s** with parameters $W$. We now add a *hyperprior* on $W$ under which the expected power of both tangential components at the $v$th voxel is determined by a hyperparameter $\alpha_v$. For notational convenience we collect the $\alpha_v$ into a vector $\boldsymbol{\alpha}$ and introduce a $d_s \times d_v$ indicator matrix $J$, with $J_{iv} = 1$ if the $i$th source is located in the $v$th voxel and 0 otherwise. Thus, each column of $J$ contains exactly two unit entries, one for each tangential component of the corresponding voxel dipole. Finally, we introduce a $d_s \times d_s$ diagonal matrix $A$ with $A_{ii} = (J\boldsymbol{\alpha})_i$. Then

$$W_{ij} \sim \mathcal{N}\left(0, A_{ii}^{-1}\right). \tag{4}$$

Thus each $\alpha_v$ sets a prior distribution on the length of the two rows in the weight matrix corresponding to source components at the $v$th voxel. As in the original ARD models, optimisation of the marginal likelihood or evidence, $\mathsf{P}\left(B \mid \boldsymbol{\alpha}, L, \Psi\right)$, with respect to the $\alpha_v$ results in a number of the hyperparameters diverging to infinity. This imposes a zero-centred delta-function prior on the corresponding row of $W$, in turn forcing the corresponding source power to vanish. It is this optimisation, then, which introduces the sparsity.

Before passing to the optimisation scheme, we summarise the model introduced above by the log joint probability it assigns to observations, pre-sources and weights (here, and below, we drop the explicit conditioning on the fixed parameters $L$ and $\Psi$)

$$\log \mathsf{P}\left(B, Z, W \mid \boldsymbol{\alpha}\right) = -\frac{1}{2}\left(N \log |2\pi\Psi| + \mathrm{Tr}\left[(B - LWZ)'\Psi^{-1}(B - LWZ)\right]\right)$$
$$-\frac{1}{2}\left(Nd_z \log(2\pi) + \mathrm{Tr}\left[Z'Z\right]\right) - \frac{1}{2}\left(d_z \log |2\pi A^{-1}| + \mathrm{Tr}\left[W'AW\right]\right) \tag{5}$$

## 3  Learning

Direct optimisation of the log marginal likelihood $\log \int dZ\ dW\ \mathsf{P}\left(B, Z, W \mid \boldsymbol{\alpha}\right)$ proves to be intractable. Instead, we adopt the "variational Bayes" (VB) framework of [3, 12]. VB is a form of the Expectation-Maximisation (EM) algorithm for maximum-likelihood estimation. Given unknown distributions $Q_z(Z)$ and $Q_w(W)$, Jensen's inequality provides a bound on the log-likelihood

$$\log \mathsf{P}\left(B \mid \boldsymbol{\alpha}\right) = \log \int dZ\ dW\ \frac{Q_z(Z)Q_w(W)}{Q_z(Z)Q_w(W)} \mathsf{P}\left(B, Z, W \mid \boldsymbol{\alpha}\right)$$
$$\geq \langle \log \mathsf{P}\left(B, Z, W \mid \boldsymbol{\alpha}\right) \rangle_{Q_z(Z)Q_w(W)} + \mathsf{H}(Q_z) + \mathsf{H}(Q_w)$$

(where $\mathsf{H}(\cdot)$ represents the Shannon entropy). This bound can then be optimised by alternate maximisations with respect to $Q_z$, $Q_w$ and the hyperparameters $\boldsymbol{\alpha}$. If, in place of the factored distribution $Q_z(Z)Q_w(W)$ we had used a joint $Q(Z, W)$, this procedure would be guaranteed to find a local maximum in the marginal likelihood (by analogy to EM). As it is, the optimisation is only approximate, but has been found to yield good maxima in a factor analysis model very similar to the one we consider here [12]. In our experiments, a slight variant of the standard VB procedure, described below, improved further on the accuracy of the solutions found.

Given estimates $Q_z^n$, $Q_w^n$ and $\boldsymbol{\alpha}^n$ at the $n$th step, the $(n+1)$th iteration is given by:

$$Q_z^{n+1}(Z) \propto \exp \langle \log \mathsf{P}\left(B, Z, W \mid \boldsymbol{\alpha}^n\right)\rangle_{Q_w^n} = \mathcal{N}\left(\Sigma_z^{n+1} \langle W \rangle'_{Q_w^n} L' \Psi^{-1} B, \Sigma_z^{n+1}\right)$$

$$\text{with} \quad \Sigma_z^{n+1} = \langle W'L'\Psi^{-1}LW + I\rangle_{Q_w^n}^{-1},$$

$$Q_w^{n+1}(W) \propto \exp \langle \log \mathsf{P}\left(B, Z, W \mid \boldsymbol{\alpha}^n\right)\rangle_{Q_z^n} = \mathcal{N}\left(\Sigma_w^{n+1} \,\mathsf{vec}\left(L'\Psi^{-1}B \langle Z'\rangle_{Q_z^{n+1}}\right), \Sigma_w^{n+1}\right);$$

$$\text{with} \quad \Sigma_w^{n+1} = \left(\langle ZZ'\rangle_{Q_z^{n+1}} \otimes L'\Psi^{-1}L + I \otimes A^n\right)^{-1},$$

$$\text{and} \quad \alpha_v^n = d_z \left(J'\mathsf{diag}\left[\langle W\rangle_{Q_w^{n+1}} \langle W\rangle'_{Q_w^{n+1}}\right]\right)_v^{-1} \left((J'\mathbf{1})_v - \alpha_v(J'\mathsf{diag}\left[\Sigma_w^{n+1}\right])_v\right).$$

where the normal distribution on $Z$ implies a normal distribution on each column $\mathbf{z}$; the distribution on $W$ is normal on $\mathsf{vec}\left(W\right)$ [2]; $\mathbf{1}$ is a vector of ones; and the $\mathsf{diag}\left[\cdot\right]$ operator returns the main diagonal of its argument as a vector.

Our experience is that better results can be obtained if the posterior expectation of $ZZ'$ in the $Q_w$ update is replaced by its value under the prior on $Z$, $NI$. This variant appears to constrain the factored posterior to remain closer to the true joint distribution. It has the additional benefit of simplifying both the notational and computational complexities of the updates (for the latter, it reduces the complexity of the inversion needed to calculate $\Sigma_w$ from $(d_s d_z)^3$ to $d_s^2$). We can then rewrite the updates into a more compact form by using this assumption, and by evaluating the expectations, to obtain

$$\Sigma_z^{n+1} = (W^{n\prime}L'\Psi^{-1}LW^n + \mathsf{Tr}\left[L'\Psi^{-1}L'\Sigma_w^n\right]I + I)^{-1} \tag{6a}$$

$$\Sigma_w^{n+1} = (NL'\Psi^{-1}L + A^n)^{-1} = (A^n)^{-1} - (A^n)^{-1}L'(N^{-1}\Psi + L(A^n)^{-1}L')^{-1}L(A^n)^{-1} \tag{6b}$$

$$W^{n+1} = \Sigma_w^{n+1}L'\Psi^{-1}BB'\Psi^{-1}LW^n\Sigma_z^{n+1} \tag{6c}$$

$$\alpha_v^{n+1} = d_z \left(J'\mathsf{diag}\left[W^{n+1}W^{n+1\prime}\right]\right)_v^{-1} \left((J'\mathbf{1})_v - \alpha_v^n(J'\mathsf{diag}\left[\Sigma_w^{n+1}\right])_v\right), \tag{6d}$$

where $W^n = \langle W\rangle_{Q_w^n}$. The use of the matrix inversion lemma in (6b) exploits the diagonality of $A$ to reduce the computational complexity of the algorithm with respect to $d_s$.

The formulae of (6) are easily implemented and recover an estimate of $W$, and thus the source correlation matrix, by iteration. The source activities can then be estimated by use of the Wiener filter (2). The updates of (6) also demonstrate an important point concerning the validity of our Gaussian model. Note that the measured data enter into the estimation procedure only through their correlation $BB'$. In other words, the hyperparameter optimisation stage of our algorithm is only being used to model the data correlation, *not their amplitudes*. As a result, the effects of incorrectly assuming a Gaussian source amplitude distribution can be expected to remain relatively benign.

## 4 Simulations

Simulation studies provide an important tool for evaluating source recovery algorithms, in that they provide "sensor" data sets for which the correct answer (i.e. the true locations and time-courses of the sources) is known. We report here the results of simulations carried out using parameters similar to those that might be encountered in realistic recordings.

### 4.1 Methods

We simulated 100 1-s-long epochs of evoked response data. The sensor configuration was taken from a real experiment: two sensor arrays, with 37 gradiometer coils each, were

located on either side of the head (see figure 1). Candidate source dipoles were located on a grid with 1 cm spacing within a hemispherical brain volume with a radius of 8 cm, to give a total of 956 possible source locations. Significant (above background) evoked activity was simulated at 5 of these locations (see figure 1a), with random dipole orientations. The evoked waveforms were similar in form to the evoked responses seen in many areas of the brain (see figure 2a), and were strongly correlated between the five sites (figure 3a). The two most lateral sites (one on each side), expressed bilateral primary sensory activation, and had identical time-courses with the shortest latency. Another lateral site, on the left side, had activity with the same waveform, but delayed by 50 ms. Two medial sites had slower and more delayed activation profiles. The dipole orientation at each site was chosen randomly in the plane parallel to the sensor tangent. Note that the amplitude distribution of these sources is strongly non-Gaussian; we will see, however, that they can be recovered successfully by the present technique despite its assumption of normality.

The simulated sensor recordings were corrupted by noise from two sources, both with Gaussian distribution. Background activity in the brain was simulated with equal power at every point on the grid of candidate sources, with a root-mean-square (RMS) amplitude 1.5 decades below that of the 5 significant sources. Although this background activity was uncorrelated between brain locations, it resulted in correlated disturbances at the magnetic sensors. Thermal noise in the sensors was uncorrelated, and had a similar magnitude (at the sensors) to that of the background noise.

The novel Bayesian estimation technique was applied to the raw simulated sensor trace rather than to epoch-averaged data. While in this simulation the evoked activity was identical in each trial, determining the correlation matrix from unaveraged data should, in the more general case, make single-trial reconstructions more accurate. Once reconstructed, the source timecourses were averaged, and are shown in figure 2. The number of pre-sources $d_z$, a free parameter in the algorithm, was set to 10. Sources associated with inverse variance hyperparameters $\alpha_i$ above a threshold (here $10^{15}$) were taken to be inactive.

For comparison, we also reconstructed sources using the vector minimum-variance adaptive beamformer approach [15]. Note that this technique, along with many other existing reconstruction methods, assumes that sources at different locations are uncorrelated and so it should not be expected to perform well under the conditions of our simulation.

## 4.2 Results

Figure 1 shows the source locations and powers reconstructed by the novel Bayesian approach developed here (b) and by the beamformer (c). The Bayesian approach identified the correct number of sources, at the correct locations and with approximately correct relative powers. By contrast, the beamformer approach, which assumes uncorrelated sources, entirely failed to locate the sources of activity.

Figure 2b shows the average evoked-response reconstruction at each of the identified source locations (with the simulated waveforms shown in panel a). The general time-course of the activities has clearly been well characterised. The time-courses estimated by the vector beamformer are shown in figure 2c. As beamformer localisation proved to be unreliable, the time-courses shown are the reconstructions at the positions of the correct (simulated) sources. Nonetheless, the strong correlations in the sources have corrupted the reconstructions. Note that the only difference between the time-courses shown in figure 2b and c is premultiplication by the estimated source correlation matrix in b.

Finally, figure 3 shows the correlation coefficient matrices for the dipole amplitude time-courses of the active sources shown in figure 2. We see that the Bayesian approach finds a reasonable approximation to the correct correlation structure. Again, however, the beamformer is unable to accurately characterise the correlation matrix.

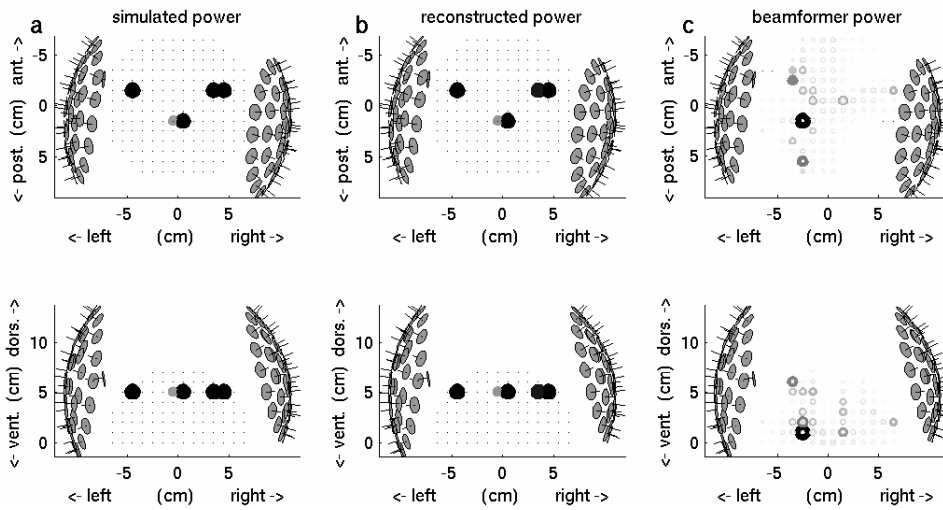

Figure 1: Reconstructed source power. Each dot represents a single voxel, the size and shade of the superimposed circles indicates the relative power of the corresponding source. Each column contains two orthogonal projections of the same source distribution: (a) simulated sources, (b) reconstruction by evidence optimisation, (c) beamformer reconstruction (powers have been compressed to make smaller sources more visible)

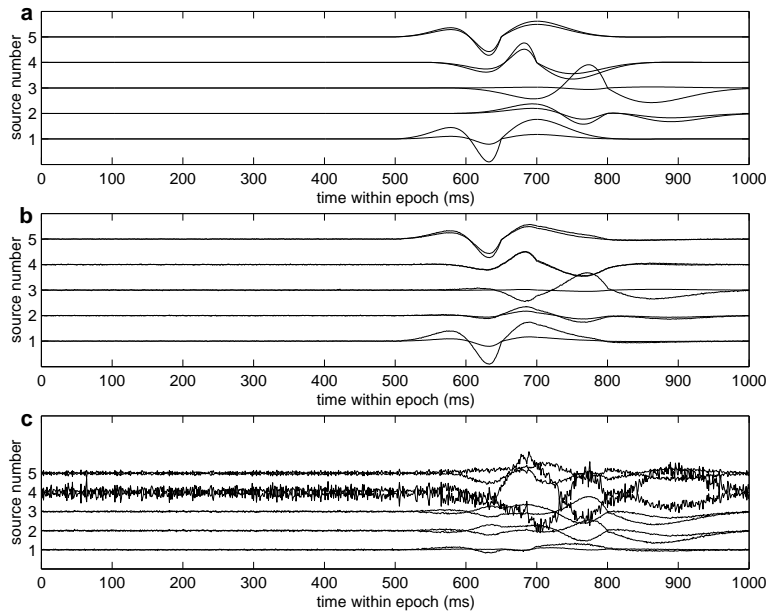

Figure 2: Source waveforms at active locations. Sources are numbered from left to right in the brain. The two traces for each location show the dipole components in two orthogonal directions. (a) simulated waveforms; (b) waveforms reconstructed by our novel algorithm; (c) waveforms reconstructed by beamforming (at the simulated locations)

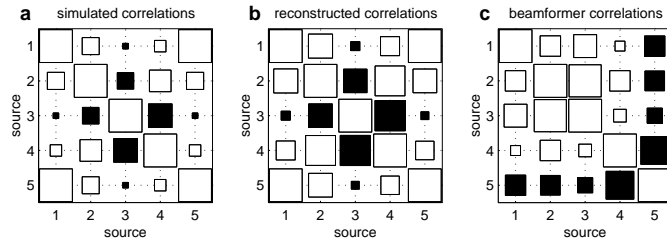

Figure 3: Source correlation coefficient matrices. Correlations were computed between epoch-averaged dipole amplitude time-courses at each location. The size of each square indicates the magnitude of the corresponding coefficient (the maximum value being 1), with whites squares positive and black squares negative. (a) simulated sources; (b) sources reconstructed by our novel algorithm; (c) sources reconstructed by beamforming.

## 5   Conclusions

We have demonstrated a novel evidence-optimisation approach to the location and reconstruction of dipole sources contributing to MEG measurements. Unlike existing methods, this new technique does not assume a correlation structure for the sources, instead estimating it from the data. As such, this approach holds great promise for high fidelity imaging of correlated magnetic activity in the brain.

### Acknowledgements

We thank Dr. Sekihara for useful discussions. This work is funded by grants from the Whitaker Foundation and from NIH (1R01004855-01A1).

## Footnotes

[1]This formulation is similar to that used in weighted minimum-norm methods, although there the weights $W$ are fixed, implying a pre-determined source correlation matrix.

[2]for a discussion of the $\mathsf{vec}$ operator and the Kronecker product $\otimes$ see e.g. [14]

### References

[1] S. Baillet, J. C. Mosher, and R. M. Leahy. *IEEE Signal Processing Magazine*, 18(6):14–30, 2001.

[2] M. Hämäläinen, R. Hari, R. Ilmoniemi, J. Knuutila, and O. V. Lounasmaa. *Rev. Mod. Phys.*, 65:413–97, 1993.

[3] H. Attias. In S. A. Solla, T. K. Leen, and K.-R. Müller, eds., *Adv. Neural Info. Processing Sys.*, vol. 12. MIT Press, 2000.

[4] A. C. Tang, B. A. Pearlmutter, N. A. Malaszenko, D. B. Phung, and B. C. Reeb. *Neural Comput.*, 14(8):1827–58, 2002.

[5] O. David, L. Garnero, D. Cosmelli, and F. J. Varela. *IEEE Trans. Biomed. Eng.*, 49(9):975–87, 2002.

[6] K. Sekihara and B. Scholz. *IEEE Trans. Biomed. Eng.*, 43(3):281–91, 1996.

[7] K. Sekihara, S. S. Nagarajan, D. Poeppel, and A. Marantz. *IEEE Trans. Biomed. Eng.*, 49(12):1234–46, 2002.

[8] C. Phillips, M. D. Rugg, and K. J. Friston. *Neuroimage*, 16(3):678–95, 2002.

[9] E. Rodriguez, N. George, J. P. Lachaux, J. Martinerie, B. Renault, and F. J. Varela. *Nature*, 397(6718):430–3, 1999.

[10] C. Bernasconi, A. von Stein, and C. Chiang. *Neuroreport*, 11(4):689–92, 2000.

[11] D. J. C. MacKay. In *ASHRAE Transactions, V.100, Pt.2*, pp. 1053–1062. ASHRAE, 1994.

[12] Z. Ghahramani and M. Beal. In S. A. Solla, T. K. Leen, and K.-R. Müller, eds., *Adv. Neural Info. Processing Sys.*, vol. 12. MIT Press, 2000.

[13] M. Sahani and J. F. Linden. In S. Becker, S. Thrun, and K. Obermayer, eds., *Adv. Neural Info. Processing Sys.*, vol. 15. MIT Press, 2003.

[14] R. A. Horn and C. R. Johnson. *Topics in Matrix Analysis*. CUP, 1991.

[15] K. Sekihara, S. S. Nagarajan, D. Poeppel, A. Marantz, and Y. Miyashita. *IEEE Trans. Biomed. Eng.*, 48(7):760–71, 2001.
